# Prediction and Semantic Association

**Thomas L. Griffiths & Mark Steyvers**
Department of Psychology
Stanford University, Stanford, CA 94305-2130
{gruffydd,msteyver}@psych.stanford.edu

## Abstract

We explore the consequences of viewing semantic association as the result of attempting to predict the concepts likely to arise in a particular context. We argue that the success of existing accounts of semantic representation comes as a result of indirectly addressing this problem, and show that a closer correspondence to human data can be obtained by taking a probabilistic approach that explicitly models the generative structure of language.

## 1 Introduction

Many cognitive capacities, such as memory and categorization, can be analyzed as systems for efficiently predicting aspects of an organism's environment [1]. Previously, such analyses have been concerned with memory for facts or the properties of objects, where the prediction task involves identifying when those facts might be needed again, or what properties novel objects might possess. However, one of the most challenging tasks people face is linguistic communication. Engaging in conversation or reading a passage of text requires retrieval of a variety of concepts from memory in response to a stream of information. This retrieval task can be facilitated by predicting which concepts are likely to be needed from their context, having efficiently abstracted and stored the cues that support these predictions.

In this paper, we examine how understanding the problem of predicting words from their context can provide insight into human semantic association, exploring the hypothesis that the association between words is at least partially affected by their statistical relationships. Several researchers have argued that semantic association can be captured using high-dimensional spatial representations, with the most prominent such approach being Latent Semantic Analysis (LSA) [5]. We will describe this procedure, which indirectly addresses the prediction problem. We will then suggest an alternative approach which explicitly models the way language is generated and show that this approach provides a better account of human word association data than LSA, although the two approaches are closely related. The great promise of this approach is that it illustrates how we might begin to relax some of the strong assumptions about language made by many corpus-based methods. We will provide an example of this, showing results from a generative model that incorporates both sequential and contextual information.

## 2  Latent Semantic Analysis

Latent Semantic Analysis addresses the prediction problem by capturing similarity in word usage: seeing a word suggests that we should expect to see other words with similar usage patterns. Given a corpus containing $W$ words and $D$ documents, the input to LSA is a $W \times D$ word-document co-occurrence matrix $\mathbf{F}$ in which $f_{wd}$ corresponds to the frequency with which word $w$ occurred in document $d$. This matrix is transformed to a matrix $\mathbf{G}$ via some function involving the term frequency $f_{wd}$ and its frequency across documents $f_{w\cdot}$. Many applications of LSA in cognitive science use the transformation

$$g_{wd} = \log\{f_{wd} + 1\}(1 - H_w) \qquad H_w = -\frac{\sum_{d=1}^{D} \frac{f_{wd}}{f_{w\cdot}} \log\{\frac{f_{wd}}{f_{w\cdot}}\}}{\log D}, \qquad (1)$$

where $H_w$ is the normalized entropy of the distribution over documents for each word. Singular value decomposition (SVD) is applied to $\mathbf{G}$ to extract a lower dimensional linear subspace that captures much of the variation in usage across words. The output of LSA is a vector for each word, locating it in the derived subspace. The association between two words is typically assessed using the cosine of the angle between their vectors, a measure that appears to produce psychologically accurate results on a variety of tasks [5]. For the tests presented in this paper, we ran LSA on a subset of the TASA corpus, which contains excerpts from texts encountered by children between first grade and the first year of college. Our subset used all $D = 37651$ documents, and the $W = 26414$ words that occurred at least ten times in the whole corpus, with stop words removed. From this we extracted a 500 dimensional representation, which we will use throughout the paper.[1]

## 3  The topic model

Latent Semantic Analysis gives results that seem consistent with human judgments and extracts information relevant to predicting words from their contexts, although it was not explicitly designed with prediction in mind. This relationship suggests that a closer correspondence to human data might be obtained by directly attempting to solve the prediction task. In this section, we outline an alternative approach that involves learning a probabilistic model of the way language is generated. One generative model that has been used to outperform LSA on information retrieval tasks views documents as being composed of sets of topics [2,4]. If we assume that the words that occur in different documents are drawn from $T$ topics, where each topic is a probability distribution over words, then we can model the distribution over words in any one document as a mixture of those topics

$$P(w_i) = \sum_{j=1}^{T} P(w_i|z_i = j)P(z_i = j) \qquad (2)$$

where $z_i$ is a latent variable indicating the topic from which the $i$th word was drawn and $P(w_i|z_i = j)$ is the probability of the $i$th word under the $j$th topic. The words likely to be used in a new context can be determined by estimating the distribution over topics for that context, corresponding to $P(z_i)$.

Intuitively, $P(w|z = j)$ indicates which words are important to a topic, while $P(z)$ is the prevalence of those topics within a document. For example, imagine a world where the only topics of conversation are love and research. We could then express

the probability distribution over words with two topics, one relating to love and the other to research. The content of the topics would be reflected in $P(w|z = j)$: the love topic would give high probability to words like JOY, PLEASURE, or HEART, while the research topic would give high probability to words like SCIENCE, MATHEMATICS, or EXPERIMENT. Whether a particular conversation concerns love, research, or the love of research would depend upon its distribution over topics, $P(z)$, which determines how these topics are mixed together in forming documents.

Having defined a generative model, learning topics becomes a statistical problem. The data consist of words $\mathbf{w} = \{w_1, \ldots, w_n\}$, where each $w_i$ belongs to some document $d_i$, as in a word-document co-occurrence matrix. For each document we have a multinomial distribution over the $T$ topics, with parameters $\theta^{(d)}$, so for a word in document $d$, $P(z_i = j) = \theta_j^{(d_i)}$. The $j$th topic is represented by a multinomial distribution over the $W$ words in the vocabulary, with parameters $\phi^{(j)}$, so $P(w_i|z_i = j) = \phi_{w_i}^{(j)}$. To make predictions about new documents, we need to assume a prior distribution on the parameters $\theta$. Existing parameter estimation algorithms make different assumptions about $\theta$, with varying results [2,4]. Here, we present a novel approach to inference in this model, using Markov chain Monte Carlo with a symmetric Dirichlet($\alpha$) prior on $\theta^{(d_i)}$ for all documents and a symmetric Dirichlet($\beta$) prior on $\phi^{(j)}$ for all topics. In this approach we do not need to explicitly represent the model parameters: we can integrate out $\theta$ and $\phi$, defining the model simply in terms of the assignments of words to topics indicated by the $z_i$.

Markov chain Monte Carlo is a procedure for obtaining samples from complicated probability distributions, allowing a Markov chain to converge to the target distribution and then drawing samples from the states of that chain (see [3]). We use Gibbs sampling, where each state is an assignment of values to the variables being sampled, and the next state is reached by sequentially sampling all variables from their distribution when conditioned on the current values of all other variables and the data. We will sample only the assignments of words to topics, $z_i$. The conditional posterior distribution for $z_i$ is given by

$$P(z_i = j|\mathbf{z}_{-i}, \mathbf{w}) \propto \frac{n_{-i,j}^{(w_i)} + \beta}{n_{-i,j}^{(\cdot)} + W\beta} \frac{n_{-i,j}^{(d_i)} + \alpha}{n_{-i,\cdot}^{(d_i)} + T\alpha} \qquad (3)$$

where $\mathbf{z}_{-i}$ is the assignment of all $z_k$ such that $k \neq i$, and $n_{-i,j}^{(w)}$ is the number of words assigned to topic $j$ that are the same as $w$, $n_{-i,j}^{(\cdot)}$ is the total number of words assigned to topic $j$, $n_{-i,j}^{(d)}$ is the number of words from document $d$ assigned to topic $j$, and $n_{-i,\cdot}^{(d)}$ is the total number of words in document $d$, all not counting the assignment of the current word $w_i$. $\alpha, \beta$ are free parameters that determine how heavily these distributions are smoothed.

We applied this algorithm to our subset of the TASA corpus, which contains $n = 5628867$ word tokens. Setting $\alpha = 0.1, \beta = 0.01$ we obtained 100 samples of 500 topics, with 10 samples from each of 10 runs with a burn-in of 1000 iterations and a lag of 100 iterations between samples.[2] Each sample consists of an assignment of every word token to a topic, giving a value to each $z_i$. A subset of the 500 topics found in a single sample are shown in Table 1. For each sample we can compute

| | | | | | |
|---|---|---|---|---|---|
| FEEL | MUSIC | BALL | SCIENCE | WORKERS | **FORCE** |
| FEELINGS | **PLAY** | GAME | STUDY | **WORK** | FORCES |
| FEELING | DANCE | TEAM | SCIENTISTS | LABOR | MOTION |
| ANGRY | PLAYS | **PLAY** | SCIENTIFIC | JOBS | BODY |
| WAY | STAGE | BASEBALL | KNOWLEDGE | WORKING | GRAVITY |
| THINK | PLAYED | FOOTBALL | **WORK** | WORKER | MASS |
| **SHOW** | BAND | PLAYERS | CHEMISTRY | WAGES | PULL |
| FEELS | AUDIENCE | GAMES | RESEARCH | FACTORY | NEWTON |
| PEOPLE | MUSICAL | PLAYING | BIOLOGY | JOB | OBJECT |
| FRIENDS | DANCING | **FIELD** | MATHEMATICS | WAGE | LAW |
| THINGS | RHYTHM | PLAYED | LABORATORY | SKILLED | DIRECTION |
| MIGHT | PLAYING | PLAYER | STUDYING | PAID | MOVING |
| HELP | THEATER | COACH | SCIENTIST | CONDITIONS | REST |
| HAPPY | DRUM | BASKETBALL | PHYSICS | PAY | FALL |
| FELT | ACTORS | SPORTS | **FIELD** | **FORCE** | ACTING |
| LOVE | **SHOW** | HIT | STUDIES | MANY | MOMENTUM |
| ANGER | BALLET | BAT | UNDERSTAND | HOURS | DISTANCE |
| BEING | ACTOR | TENNIS | STUDIED | EMPLOYMENT | GRAVITATIONAL |
| WAYS | DRAMA | TEAMS | SCIENCES | EMPLOYED | PUSH |
| FEAR | SONG | SOCCER | MANY | EMPLOYERS | VELOCITY |

Table 1: Each column shows the 20 most probable words in one of the 500 topics obtained from a single sample. The organization of the columns and use of boldface displays the way in which polysemy is captured by the model.

the posterior predictive distribution (and posterior mean for $\phi^{(j)}$):

$$P(w|z=j,\mathbf{z},\mathbf{w}) = \int P(w|z=j,\phi^{(j)})P(\phi^{(j)}|\mathbf{z},\mathbf{w})\ d\phi^{(j)} = \frac{n_j^{(w)}+\beta}{n_j^{(\cdot)}+W\beta} \quad (4)$$

# 4 Predicting word association

We used both LSA and the topic model to predict the association between pairs of words, comparing these results with human word association norms collected by Nelson, McEvoy and Schreiber [7]. These word association norms were established by presenting a large number of participants with a cue word and asking them to name an associated word in response. A total of 4544 of the words in these norms appear in the set of 26414 taken from the TASA corpus.

## 4.1 Latent Semantic Analysis

In LSA, the association between two words is usually measured using the cosine of the angle between their vectors. We ordered the associates of each word in the norms by their frequencies, making the first associate the word most commonly given as a response to the cue. For example, the first associate of NEURON is BRAIN. We evaluated the cosine between each word and the other 4543 words in the norms, and then computed the rank of the cosine of each of the first ten associates, or all of the associates for words with less than ten. The results are shown in Figure 1. Small ranks indicate better performance, with a rank of one meaning that the target word had the highest cosine. The median rank of the first associate was 32, and LSA correctly predicted the first associate for 507 of the 4544 words.

## 4.2 The topic model

The probabilistic nature of the topic model makes it easy to predict the words likely to occur in a particular context. If we have seen word $w_1$ in a document, then we can determine the probability that word $w_2$ occurs in that document by computing $P(w_2|w_1)$. The generative model allows documents to contain multiple topics, which

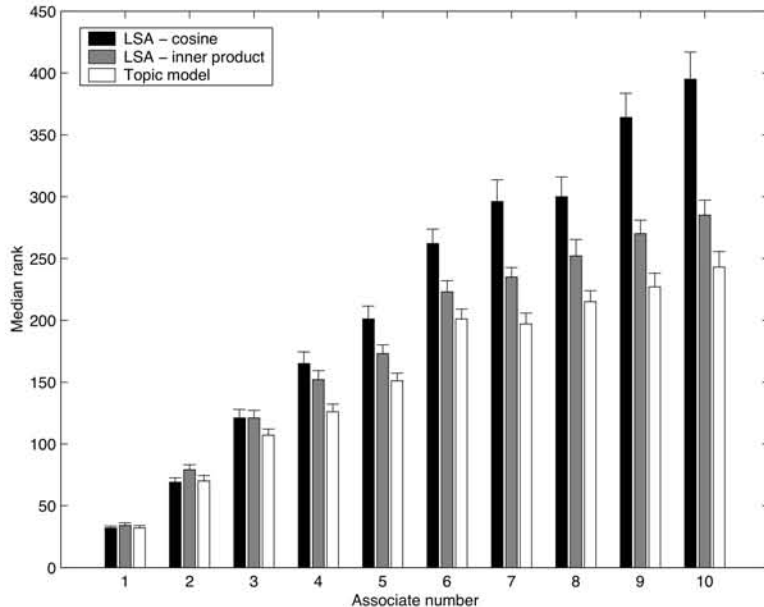

Figure 1: Performance of different methods of prediction on the word association task. Error bars show one standard error, estimated with 1000 bootstrap samples.

is extremely important to capturing the complexity of large collections of words and computing the probability of complete documents. However, when comparing individual words it is more effective to assume that they both come from a single topic. This assumption gives us

$$P_1(w_2|w_1) = \sum_z P(w_2|z)P(z|w_1) \qquad P(z|w_1) \propto P(w_1|z)P(z) \qquad (5)$$

where we use Equation 4 for $P(w|z)$ and $P(z)$ is uniform, consistent with the symmetric prior on $\theta$, and the subscript in $P_1(w_2|w_1)$ indicates the restriction to a single topic. This estimate can be computed for each sample separately, and an overall estimate obtained by averaging over samples. We computed $P_1(w_2|w_1)$ for the 4544 words in the norms, and then assessed the rank of the associates in the resulting distribution using the same procedure as for LSA. The results are shown in Figure 1. The median rank for the first associate was 32, with 585 of the 4544 first associates exactly correct. The probabilistic model performed better than LSA, with the improved performance becoming more apparent for the later associates.

### 4.3 Discussion

The central problem in modeling semantic association is capturing the interaction between word frequency and similarity of word usage. Word frequency is an important factor in a variety of cognitive tasks, and one reason for its importance is its predictive utility. A higher observed frequency means that a word should be predicted to occur more often. However, this effect of frequency should be tempered by the relationship between a word and its semantic context. The success of the topic model is a consequence of naturally combining frequency information with semantic similarity: when a word is very diagnostic of a small number of topics, semantic context is used in prediction. Otherwise, word frequency plays a larger role.

The effect of word frequency in the topic model can be seen in the rank-order correlation of the predicted ranks of the first associates with the ranks predicted by word frequency alone, which is $\rho = 0.49$. In contrast, the cosine is used in LSA because it explicitly removes the effect of word frequency, with the corresponding correlation being $\rho = -0.01$. The cosine is purely a measure of semantic similarity, which is useful in situations where word frequency is misleading, such as in tests of English fluency or other linguistic tasks, but not necessarily consistent with human performance. This measure was based in the origins of LSA in information retrieval, but other measures that do incorporate word frequency have been used for modeling psychological data. We consider one such measure in the next section.

## 5    Relating LSA and the topic model

The decomposition of a word-document co-occurrence matrix provided by the topic model can be written in a matrix form similar to that of LSA. Given a word-document co-occurrence matrix $\mathbf{F}$, we can convert the columns into empirical estimates of the distribution over words in each document by dividing each column by its sum. Calling this matrix $\mathbf{P}$, the topic model approximates it with the non-negative matrix factorization $\mathbf{P} \approx \boldsymbol{\phi}\boldsymbol{\theta}$, where column $j$ of $\boldsymbol{\phi}$ gives $\phi^{(j)}$, and column $d$ of $\boldsymbol{\theta}$ gives $\theta^{(d)}$. The inner product matrix $\mathbf{PP}^T$ is proportional to the empirical estimate of the joint distribution over words $P(w_1, w_2)$. We can write $\mathbf{PP}^T \approx \boldsymbol{\phi}\boldsymbol{\theta}\boldsymbol{\theta}^T\boldsymbol{\phi}^T$, corresponding to $P(w_1, w_2) = \sum_{z_1, z_2} P(w_1|z_1)P(w_2|z_2)P(z_1, z_2)$, with $\boldsymbol{\theta}\boldsymbol{\theta}^T$ an empirical estimate of $P(z_1, z_2)$. The theoretical distribution for $P(z_1, z_2)$ is proportional to $\mathbf{I} + \alpha$, where $\mathbf{I}$ is the identity matrix, so $\boldsymbol{\theta}\boldsymbol{\theta}^T$ should be close to diagonal. The single topic assumption removes the off-diagonal elements, replacing $\boldsymbol{\theta}\boldsymbol{\theta}^T$ with $\mathbf{I}$ to give $P_1(w_1, w_2) \propto \boldsymbol{\phi}\boldsymbol{\phi}^T$.

By comparison, LSA transforms $\mathbf{F}$ to a matrix $\mathbf{G}$ via Equation 1, then the SVD gives $\mathbf{G} \approx \mathbf{UDV}^T$ for some low-rank diagonal $\mathbf{D}$. The locations of the words along the extracted dimensions are $\mathbf{X} = \mathbf{UD}$. If the column sums do not vary extensively, the empirical estimate of the joint distribution over words specified by the entries in $\mathbf{G}$ will be approximately $P(w_1, w_2) \propto \mathbf{GG}^T$. The properties of the SVD guarantee that $\mathbf{XX}^T$, the matrix of inner products among the word vectors, is the best low-rank approximation to $\mathbf{GG}^T$ in terms of squared error. The transformations in Equation 1 are intended to reduce the effects of word frequency in the resulting representation, making $\mathbf{XX}^T$ more similar to $\boldsymbol{\phi}\boldsymbol{\phi}^T$.

We used the inner product between word vectors to predict the word association norms, exactly as for the cosine. The results are shown in Figure 1. The inner product initially shows worse performance than the cosine, with a median rank of 34 for the first associate and 500 exactly correct, but performs better for later associates. The rank-order correlation with the predictions of word frequency for the first associate was $\rho = 0.46$, similar to that for the topic model. The rank-order correlation between the ranks given by the inner product and the topic model was $\rho = 0.81$, while the cosine and the topic model correlate at $\rho = 0.69$. The inner product and $P_1(w_2|w_1)$ in the topic model seem to give quite similar results, despite being obtained by very different procedures. This similarity is emphasized by choosing to assess the models with separate ranks for each cue word, since this measure does not discriminate between joint and conditional probabilities. While the inner product is related to the joint probability of $w_1$ and $w_2$, $P_1(w_2|w_1)$ is a conditional probability and thus allows reasonable comparisons of the probability of $w_2$ across choices of $w_1$, as well as having properties like asymmetry that are exhibited by word association.

| "syntax" | | | | "semantics" | |
|---|---|---|---|---|---|
| HE | ON | BE | SAID | MAP | DOCTOR |
| YOU | AT | MAKE | ASKED | NORTH | PATIENT |
| THEY | INTO | GET | THOUGHT | EARTH | HEALTH |
| I | FROM | HAVE | TOLD | SOUTH | HOSPITAL |
| SHE | WITH | GO | SAYS | POLE | MEDICAL |
| WE | THROUGH | TAKE | MEANS | MAPS | CARE |
| IT | OVER | DO | CALLED | EQUATOR | PATIENTS |
| PEOPLE | AROUND | FIND | CRIED | WEST | NURSE |
| EVERYONE | AGAINST | USE | SHOWS | LINES | DOCTORS |
| OTHERS | ACROSS | SEE | ANSWERED | EAST | MEDICINE |
| SCIENTISTS | UPON | HELP | TELLS | AUSTRALIA | NURSING |
| SOMEONE | TOWARD | KEEP | REPLIED | GLOBE | TREATMENT |
| WHO | UNDER | GIVE | SHOUTED | POLES | NURSES |
| NOBODY | ALONG | LOOK | EXPLAINED | HEMISPHERE | PHYSICIAN |
| ONE | NEAR | COME | LAUGHED | LATITUDE | HOSPITALS |
| SOMETHING | BEHIND | WORK | MEANT | PLACES | DR |
| ANYONE | OFF | MOVE | WROTE | LAND | SICK |
| EVERYBODY | ABOVE | LIVE | SHOWED | WORLD | ASSISTANT |
| SOME | DOWN | EAT | BELIEVED | COMPASS | EMERGENCY |
| THEN | BEFORE | BECOME | WHISPERED | CONTINENTS | PRACTICE |

Table 2: Each column shows the 20 most probable words in one of the 48 "syntactic" states of the hidden Markov model (four columns on the left) or one of the 150 "semantic" topics (two columns on the right) obtained from a single sample.

# 6    Exploring more complex generative models

The topic model, which explicitly addresses the problem of predicting words from their contexts, seems to show a closer correspondence to human word association than LSA. A major consequence of this analysis is the possibility that we may be able to gain insight into some of the associative aspects of human semantic memory by exploring statistical solutions to this prediction problem. In particular, it may be possible to develop more sophisticated generative models of language that can capture some of the important linguistic distinctions that influence our processing of words. The close relationship between LSA and the topic model makes the latter a good starting point for an exploration of semantic association, but perhaps the greatest potential of the statistical approach is that it illustrates how we might go about relaxing some of the strong assumptions made by both of these models.

One such assumption is the treatment of a document as a "bag of words", in which sequential information is irrelevant. Semantic information is likely to influence only a small subset of the words used in a particular context, with the majority of the words playing functional syntactic roles that are consistet across contexts. Syntax is just as important as semantics for predicting words, and may be an effective means of deciding if a word is context-dependent. In a preliminary exploration of the consequences of combining syntax and semantics in a generative model for language, we applied a simple model combining the syntactic structure of a hidden Markov model (HMM) with the semantic structure of the topic model. Specifically, we used a third-order HMM with 50 states in which one state marked the start or end of a sentence, 48 states each emitted words from a different multinomial distribution, and one state emitted words from a document-dependent multinomial distribution corresponding to the topic model with $T = 150$. We estimated parameters for this model using Gibbs sampling, integrating out the parameters for both the HMM and the topic model and sampling a state and a topic for each of the 11821091 word tokens in the corpus.[3] Some of the state and topic distributions from a single sample after 1000 iterations are shown in Table 2. The states of the HMM accurately picked out many of the functional classes of English syntax, while the state corresponding to the topic model was used to capture the context-specific distributions over nouns.

Combining the topic model with the HMM seems to have advantages for both: no function words are absorbed into the topics, and the HMM does not need to deal with the context-specific variation in nouns. The model also seems to do a good job of generating topic-specific text – we can clamp the distribution over topics to pick out those of interest, and then use the model to generate phrases. For example, we can generate phrases on the topics of research ("the chief wicked selection of research in the big months", "astronomy peered upon your scientist's door", or "anatomy established with principles expected in biology"), language ("he expressly wanted that better vowel"), and the law ("but the crime had been severely polite and confused", or "custody on enforcement rights is plentiful"). While these phrases are somewhat nonsensical, they are certainly topical.

## 7    Conclusion

Viewing memory and categorization as systems involved in the efficient prediction of an organism's environment can provide insight into these cognitive capacities. Likewise, it is possible to learn about human semantic association by considering the problem of predicting words from their contexts. Latent Semantic Analysis addresses this problem, and provides a good account of human semantic association. Here, we have shown that a closer correspondence to human data can be obtained by taking a probabilistic approach that explicitly models the generative structure of language, consistent with the hypothesis that the association between words reflects their probabilistic relationships. The great promise of this approach is the potential to explore how more sophisticated statistical models of language, such as those incorporating both syntax and semantics, might help us understand cognition.

**Acknowledgments**

This work was generously supported by the NTT Communications Sciences Laboratories. We used Mersenne Twister code written by Shawn Cokus, and are grateful to Touchstone Applied Science Associates for making available the TASA corpus, and to Josh Tenenbaum for extensive discussions on this topic.

## Footnotes

[1]The dimensionality of the representation is an important parameter for both models in this paper. LSA performed best on the word association task with around 500 dimensions, so we used the same dimensionality for the topic model.

[2]Random numbers were generated with the Mersenne Twister, which has an extremely deep period [6]. For each run, the initial state of the Markov chain was found using an on-line version of Equation 3.

[3]This larger number is a result of including low frequency and stop words.

## References

[1] J. R. Anderson. *The Adaptive Character of Thought.* Erlbaum, Hillsdale, NJ, 1990.

[2] D. M. Blei, A. Y. Ng, and M. I. Jordan. Latent Dirichlet allocation. In T. G. Dietterich, S. Becker, and Z. Ghahramani, eds, *Advances in Neural Information Processing Systems 14*, 2002.

[3] W. R. Gilks, S. Richardson, and D. J. Spiegelhalter, eds. *Markov Chain Monte Carlo in Practice.* Chapman and Hall, Suffolk, 1996.

[4] T. Hofmann. Probabilistic Latent Semantic Indexing. In *Proceedings of the Twenty-Second Annual International SIGIR Conference*, 1999.

[5] T. K. Landauer and S. T. Dumais. A solution to Plato's problem: The Latent Semantic Analysis theory of acquisition, induction, and representation of knowledge. *Psychological Review*, 104:211–240, 1997.

[6] M. Matsumoto and T. Nishimura. Mersenne twister: A 623-dimensionally equidistributed uniform pseudorandom number generator. *ACM Transactions on Modeling and Computer Simulation*, 8:3–30, 1998.

[7] D. L. Nelson, C. L. McEvoy, and T. A. Schreiber. The University of South Florida word association norms. *http://www.usf.edu/FreeAssociation*, 1999.
